# Effects of Stimulus Type and of Error-Correcting Code Design on BCI Speller Performance

**Jeremy Hill**[1]      **Jason Farquhar**[2]      **Suzanne Martens**[1]

**Felix Bießmann**[1,3]      **Bernhard Schölkopf**[1]

[1]Max Planck Institute for Biological Cybernetics
`{firstname.lastname}@tuebingen.mpg.de`

[2]NICI, Radboud University, Nijmegen, The Netherlands
`J.Farquhar@nici.ru.nl`

[3]Dept of Computer Science, TU Berlin, Germany

## Abstract

From an information-theoretic perspective, a noisy transmission system such as a visual Brain-Computer Interface (BCI) speller could benefit from the use of error-correcting codes. However, optimizing the code solely according to the maximal minimum-Hamming-distance criterion tends to lead to an overall increase in target frequency of target stimuli, and hence a significantly reduced average target-to-target interval (TTI), leading to difficulties in classifying the individual event-related potentials (ERPs) due to overlap and refractory effects. Clearly any change to the stimulus setup must also respect the possible psychophysiological consequences. Here we report new EEG data from experiments in which we explore stimulus types and codebooks in a within-subject design, finding an interaction between the two factors. Our data demonstrate that the traditional, row-column code has particular spatial properties that lead to better performance than one would expect from its TTIs and Hamming-distances alone, but nonetheless error-correcting codes can improve performance provided the right stimulus type is used.

## 1 Introduction

The Farwell-Donchin speller [4], also known as the "P300 speller," is a Brain-Computer Interface which enables users to spell words provided that they can see sufficiently well. This BCI determines the intent of the user by recording and classifying his electroencephalogram (EEG) in response to controlled stimulus presentations. Figure 1 shows a general P300 speller scheme. The stimuli are intensifications of a number of letters which are organized in a grid and displayed on a screen. In a standard setup, the rows and columns of the grid flash in a random order. The intensification of the row or column containing the letter that the user wants to communicate is a target in a stimulus sequence and induces a different brain response than the intensification of the other rows and columns (the non-targets). In particular, targets and non-targets are expected to elicit certain event-related potential (ERP) components, such as the so-called P300, to different extents. By classifying the epochs (i.e. the EEG segments following each stimulus event) into targets and non-targets, the target row and column can be predicted, resulting in the identification of the letter of interest.

The classification process in the speller can be considered a noisy communication channel where the sequence of EEG epochs is a modulated version of a bit string denoting the user's desired letter.

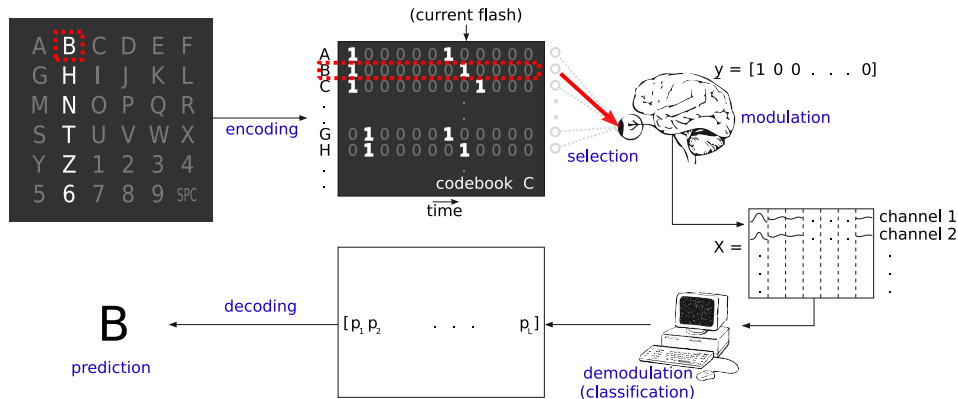

Figure 1: Schematic of the visual speller system, illustrating the relationship between the spatial pattern of flashes and one possible codebook for letter transmission (flash rows then columns).

These bit strings or codewords form the rows of a binary codebook C, a matrix in which a 1 at position $(i, j)$ means the letter corresponding to row $i$ flashed at time-step $j$, and a 0 indicates that it did not. The standard row-column code, in which exactly one row or exactly one column flashes at any one time, will be denoted **RC**. It is illustrated in figure 1.

A classifier decodes the transmitted information into an output bit string. In practice, the poor signal-to-noise ratio of the ERPs hampers accurate classification of the epochs, so the output bit string may differ from the transmitted bit string (decoding error). Also, the transmitted string may differ from the corresponding row in the codebook due to modulation error, for example if the user lost his attention and missed a stimulus event. Coding theory tells us that we can detect and correct transmission and decoding errors by adding redundancy to the transmitted bit string. The Hamming distance $d$ is the number of bit positions that differ between two rows in a codebook. The minimum Hamming distance $d_{\min}$ of all pairs of codewords is related to the error correcting abilities of the code by $e = (d_{\min} - 1)/2$, where $e$ is the maximum number of errors that a code can guarantee to correct [9]. In general, we find the mean Hamming distance within a given codebook to be a rough predictor of that codebook's performance.

In the standard approach, redundancy is added by repeating the flashing of all rows and columns $R$ times. This leads to $d = 4R$ between two letters not in the same row or column and $d_{\min} = 2R$ between two letters in the same row or column. The **RC** code is a poor code in terms of minimum Hamming distance: to encode 36 different letters in 12 bits, $d_{\min} = 4$ is possible, and the achievable $d_{\min}$ increases supra-linearly with the total code length $L$ (for example, $d_{\min} = 10$ is possible in $L = 24$ bits, the time taken for $R = 2$ repeats of the **RC** code).

However, the codes with a larger $d_{\min}$ are characterized by an increased *weight* compared to the **RC** code, i.e. the number of 1's per bitstring is larger. As target stimulus events occur more frequently overall, the expected target-to-target interval (TTI) decreases. One cannot approach codebook optimization, therefore, without asking what effect this might have on the signals we are trying to measure and classify, namely the ERPs in response to the stimulus events.

The speller was originally derived from an "oddball" paradigm, in which subjects are presented with a repetitive sequence of events, some of which are targets requiring a different response from the (more frequent) non-targets. The targets are expected to evoke a larger P300 than the non-targets. It was generally accepted that the amplitude of the target P300 decreases when the percentage of targets increases [3, 11]. However, more recently, it was suggested that the observed tendency of the P300 amplitude (as measured by averaging over many targets) to decrease with increased target probability may in fact be attributed to greater prevalence of shorter target-to-target intervals (TTI) [6] rather than an overall effect of target frequency per se. In a different type of paradigm using only targets, it was shown that at TTIs smaller than about 1 second, the P300 amplitude is significantly decreased due to refractory effects [15]. Typical stimulus onset asynchronies (SOAs) in the oddball paradigm are in the order of seconds since the P300 component shows up somewhere between 200 and 800 msec[12]. In spellers, small SOAs of about 100 msec are often used [8, 13] in order to

achieve high information transfer rates. Consequently, one can expect a significant ERP overlap into the epoch following a target epoch, and since row flashes are often randomly mixed in with column flashes, different targets may experience very different TTIs. For a $6 \times 6$ grid, the TTI ranges from $1 \times$SOA to $20 \times$SOA, so targets may suffer to varying degrees from any refractory and overlap effects.

In order to quantify the detrimental effects of short TTI we examined data from the two subjects in dataset IIa+b from the BCI Competition III[2]. Following the classification procedures described in section 3.3, we estimated classification performance on the individual epochs of both data sets by 10-fold cross-validation within each subject's data set. Binary (target versus non-target) classification results were separated according to the time since the previous target (TPT)—for the targets this distance measure is equivalent to the TTI. The left panel of fig 4 shows the average classification error as a function of TPT (averaged across both subjects—both subjects show the same qualitative effect). Evidently, the target epochs with a TPT$<$ 0.5 sec display a classification accuracy that approximates chance performance. Consequently, the target epochs with TPT$<$ 0.5 sec, constituting about 20% of all target epochs in a **RC** code, do not appear to be useful for transmission [10].

Clearly, there is a potential conflict between information-theoretic factors, which favour increasing the minimum Hamming distance and hence the overall proportion of target stimuli, and the detrimental psychophysiological effects of doing so.

In [7] we explored this trade-off to see whether an optimal compromise could be found. We initially built a generative model of the BCI system, using the competition data illustrated in figure 4, and then used this model to guide the generation and selection of speller code books. The results were not unequivocally successful: though we were able to show effects of both TTIs and of the Hamming distances in our codebooks, our optimized codebook performed no better than the row-column code for the standard flash stimulus. However, our series of experiments involved another kind of stimulus, and the effect of our codebook manipulation was found to interact with the kind of stimulus used.

The purpose of the current paper is two-fold:

1. to present new data which ilustrate the stimulus/codebook interaction more clearly, and demonstrate the advantage to be gained by the correct choice of stimulus together with an error-correcting code.

2. to present evidence for another effect, which we had not previously considered in modelling our subjects' responses, which may explain why row-column codes perform better than expected: specifically, the spatial contiguity of rows and columns.

## 2 Decoding Framework

### 2.1 Probabilistic Approach to Classification and Decoding

We assume an $N$-letter alphabet $\Gamma$ and an $N$-letter by $L$-bit codebook C. The basic demodulation and decoding procedure consists of finding the letter $\hat{T}$ among the possible letters $t \in \Gamma$ showing the largest probability $\Pr(t|X)$ of being the target letter $T$, given C and the measured brain signals $X = [x_1, \ldots, x_L]$, i.e.,

$$\hat{T} = \underset{t \in \Gamma}{\operatorname{argmax}} \Pr(t|X) = \underset{t \in \Gamma}{\operatorname{argmax}} \frac{\Pr(X|t) \Pr(t)}{\Pr(X)} , \qquad (1)$$

where the second equality follows from Bayes' rule. A simple approach to decoding is to treat the individual binary epochs, with binary labels $\underline{c} = (C_{t1} \ldots C_{tL})$, as independent. This allows us to factor $\Pr(X|t)$ into per-epoch probabilities $\Pr(x_j|\underline{c})$ for epoch indices $j = 1 \ldots L$, to give

$$\Pr(t|X) = \frac{\Pr(t)}{\Pr(X)} \prod_{j=1}^{L} \Pr(x_j|\underline{c}) = \frac{\Pr(t)}{\Pr(X)} \prod_{j=1}^{L} \frac{\Pr(C_{tj}|x_j) \Pr(x_j)}{\Pr(C_{tj})} = f_t(X) , \qquad (2)$$

where the second equality again follows from Bayes' rule.

This form of Bayesian decoding [5] forms the basis for our decoding scheme. We train a probabilistic discriminative classifier, in particular a linear logistic regression (LR) classifier [1, pp82-85], to

estimate $\Pr\left(C_{tj}|x_j\right) = p_j$ in (2). As a result, we can obtain estimates of the probability $\Pr\left(t|X\right)$ that a particular letter $t$ corresponds to the user-selected codeword. Note that for decoding purposes the terms $\Pr\left(X\right)$ and $\Pr\left(x_j\right)$ can be ignored as they are independent of $t$. Furthermore, the product $\prod_j \Pr\left(C_{tj}\right)$ depends only on the positive-class prior of the binary classifier, $\Pr\left(+\right)$. In fact, it is easy to show that during decoding this term cancels out the effect of the binary prior, which may therefore be set arbitrarily without affecting the decisions made by our decoder. The simplest thing to do is to train classifiers with $\Pr\left(+\right) = 0.5$, in which case the denominator term is constant for all $t$.

### 2.1.1 Codebook Optimization

We used a simple model of subjects' responses in each epoch in order to estimate the probability of making a prediction error with the above decoding method. We used it to compute the *codebook loss*, which is the sum of error probabilities, weighted by the probability of transmission of each letter. This loss function was then minimized in order to obtain an optimized codebook.

Note that this approach is not a direct attempt to tackle the tendency for the performance of the binary target-vs-nontarget classifier to deteriorate when TTI is short (although this would surely be a promising alternative strategy). Instead, we take a "normal" classifier, as susceptible to short-TTI effects as classifiers in any other study, but try to estimate the negative impact of such effects, and then find the best trade-off between avoiding short TTIs on the one hand, and having large Hamming distances on the other hand.

Since our optimization did not result in a decisive gain in performance, we do not wish to emphasize the details of the optimization methods here. However, for further details see the supplementary material, or our tech report [7]. For the purposes of the current paper it is the properties of the resulting codebooks that are important, rather than the precise criterion according to which they are considered theoretically optimal. The codebooks themselves are described in section 3.1 and given in full in the supplementary material.

## 3 EEG Experiments

We implemented a Farwell/Donchin-style speller, using a $6 \times 6$ grid of alphanumeric characters, presented via an LCD monitor on a desk in a quiet office. Subjects each performed a single 3-hour session during which their EEG signals were measured using a QuickAmp system (BrainProducts GmbH) in combination with an Electro-Cap. The equipment was set up to measure 58 channels of EEG, one horizontal EOG at the left eye, one bipolar vertical EOG signal, and a synchronization signal from a light sensor attached to the display, all sampled at 250 Hz. We present results from 6 healthy subjects in their 20s and 30s (5 male, 1 female).

Two factors were compared in a fully within-subject design: codebook and stimulus. These are described in the next two subsections.

### 3.1 Codebook Comparison

In total, we explored 5 different stimulus codes:

1. $\mathbf{RC}_{\mathrm{mix}}$: the 12-bit row-column code, with the 12 bits randomly permuted in time (row events mixed up randomly between column events) as in the competition data [2].

2. $\mathbf{RC}_{\mathrm{sep}}$: the 12-bit row-column code, where the 6 rows are intensified in random order, and then the 6 columns in random order.

3. $\mathbf{RC}_*$: this code was generated by taking code $\mathbf{RC}_{\mathrm{sep}}$ and randomizing the assignment between codewords and letters. Thus, the TTI and Hamming-distance content of the codebook remained identical to $\mathbf{RC}_{\mathrm{sep}}$, but the spatial contiguity of the stimulus events was broken: that is to say, it was no longer a coherent row or column that flashed during any one epoch, but rather a collection of 6 apparently randomly scattered letters. However, if a subject were to have "tunnel vision" and be unable to see any letters other than the target, this would be exactly equivalent to $\mathbf{RC}_{\mathrm{sep}}$. As we shall see, for the purposes of the speller, our subjects do not have tunnel vision.

| code | L | $d_{\min}$ | E($d$) | E(TTI) | E(#11) | Pr$(1)$ | $\mathcal{L}$ |
|---|---|---|---|---|---|---|---|
| **RC**$_{\mathrm{mix}}$ $\times 2$ | 24 | 4 | 6.9 | 5.4 | 0.4 | 0.17 | 0.60 |
| **RC**$_{\mathrm{sep}}$ $\times 2$ | 24 | 4 | 6.9 | 6.0 | 0.1 | 0.17 | 0.56 |
| **RC**$_{*}$ $\times 2$ | 24 | 4 | 6.9 | 6.0 | 0.1 | 0.17 | 0.56 |
| **D10** | 24 | 10 | 11.5 | 2.5 | 3.1 | 0.38 | 0.54 |
| **D8**$_{\mathrm{opt}}$ | 24 | 8 | 10.7 | 3.1 | 0.0 | 0.32 | 0.44 |

Table 1: Summary statistics for the 24-bit versions of the 5 codebooks used. E(#11) means the average number of consecutive target letters per codeword, and Pr$(1)$ the proportion of targets. $\mathcal{L}$ is our estimated probability of an error, according to the model (see supplementary material or [7]).

4. **D10**: a 24-bit code with the largest minimum Hamming distance we could achieve ($d_{\min} = 10$). To make it, our heuristic for codeword selection was to pick the codeword with the largest minimum distance between it and all previously selected codewords. A large number of candidate codebooks were generated this way, and the criteria for scoring a completed codebook were (first) $d_{\min}$ and (second, to select among a large number of $d_{\min} = 10$ candidates) the lowest number of consecutive targets.

5. **D8**$_{\mathrm{opt}}$: a 24-bit code optimized according to our model. The heuristic for greedy codeword selection was the mean pairwise codebook loss w.r.t. previously selected codebook entries, and the final scoring criterion was our overall codebook loss function.

## 3.2 Stimulus Comparison

Two stimulus conditions were compared. In both conditions, stimulus events were repeated with a stimulus onset asynchrony (SOA) of 167 msec, which as close as our hardware could come to recreating the 175-msec SOA of competition III dataset II.

**Flashes**: grey letters presented on a black background were flashed in a conventional manner, being intensified to white for 33 msec (two video frames). An example is illustrated in the inset of the left panel of figure 2.

**Flips**: each letter was superimposed on a small grey rectangle whose initial orientation was either horizontal or vertical (randomly determined for each letter). Instead of the letter flashing, the rectangle flipped its orientation instantaneously by 90°. An example is illustrated in the inset of the right panel of figure 2. Our previous experiments had led us to conclude that many subjects perform significantly better with this stimulus, and find it more pleasant, than the flash. As we shall see, our results from this stimulus condition support this finding, and indicate a potentially useful interaction between stimulus type and codebook design.

## 3.3 Experimental Procedure

The experiment was divided into blocks, each block containing 20 trials with short (2–4 second) rest pauses between trials. Each trial began with a red box which indicated to the subject which letter (randomly chosen on each trial) they should attend to—this cue came on for a second, and was removed 1 second before the start of the stimulus sequence. Subjects were instructed to count the stimulus events at the target location, and not to blink, move or swallow during the sequence. The sequence consisted of $L = 72$ stimulus events, their spatio-temporal arrangement being determined by one of the five code conditions. The 12-bit **RC** codes were repeated six times in order to make the length up to $L = 72$ (re-randomizing the row and column order on each repetition) and the 24-bit optimized codes were repeated three times (reassigning the codewords between repetitions to ensure maximal gap between targets at the end of one repetition and the beginning of the next) likewise to ensure a total code length of 72 bits.

Each of the 5 code conditions occurred 4 times per block, the order of their occurrence being randomized. For a given block, the stimulus condition was held constant, but the stimulus type was alternated between blocks. In total, each subject performed 16 blocks. Thus, in each of the 10 stimulus $\times$ code conditions, there were a total of 32 letter presentations or 2304 stimulus events.

### 3.3.1 Online Verification

Subjects did not receive feedback at the end of each trial. However, at the end of the experiment, we gave the subject the opportunity to perform free-spelling in order to validate the system's performance: we asked each subject whether they would prefer to spell with flips or flashes, and loaded a classifier trained on all data from their preferred stimulus type into the system. Using the 72-bit codebooks, all subjects were able to spell 5-15 letters with online performance ranging from 90 to 100%. Our data analysis below is restricted to leave-one-letter-out offline performance, excluding the free-spelled letters.

### 3.4 Data Analysis

The 60-channel data, sampled at 250 Hz, were band-pass filtered between 0.1 and 8 Hz using a FIR filter. The data were then cut into 600-msec (150-sample) epochs time-locked to the stimulus events, and these were downsampled to 25 Hz. The data were then whitened in 60-dimensional sensor space (by applying a symmetric spatial filtering matrix equal to the matrix-square-root of the data covariance matrix, computed across all training trials and time-samples). Finally a linear LR classifier was applied [1, pp82-85]. The classifier's regularization hyperparameter $C$ was found by 10-fold cross-validation within the training set..

Offline letter classification performance was assessed by a leave-one-letter-out procedure: for a given code condition, each of the 32 letters was considered in turn, and a probabilistic prediction was made of its binary epoch labels using the above procedure trained only on epochs from the other 31 letters. These probabilities were combined using the decoding scheme described in section 2.1 and a prediction was made of the transmitted letter. We varied the number of consecutive epochs of the test letter that the decoder was allowed to use, from the minimum (12 or 24) up to the maximum 72. For each epoch of the left-out letter, we also recorded whether the binary classifier correctly classified the epoch as a target or non-target.

## 4 Results and Discussion

Estimates of 36-class letter prediction performance are shown in figures 2 (averaged across subjects, as a function of codeword length) and 3 (for each individual subject, presenting only the results for 24-bit codewords). The performance of the binary classifier on individual epochs is shown in figure 4.

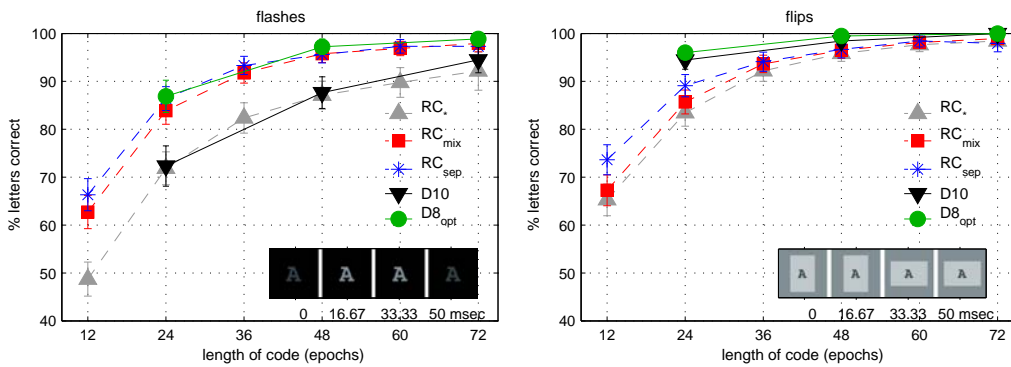

Figure 2: Offline (leave-one-letter-out) 36-class prediction performance as a function of codeword length (i.e. the number of consecutive epochs of the left-out letter that were used to make a prediction). Performance values (and standard-error bar heights) are averaged across the 6 subjects.

Our results indicated the following effects:

1. Using the Donchin flash stimulus, the deleterious effects of short TTIs were clear to see: **D10** performed far worse than the other codes despite its larger Hamming distances. In both stimulus conditions, the averaged plots of figure 2 indicate that $RC_{mix}$ may also be

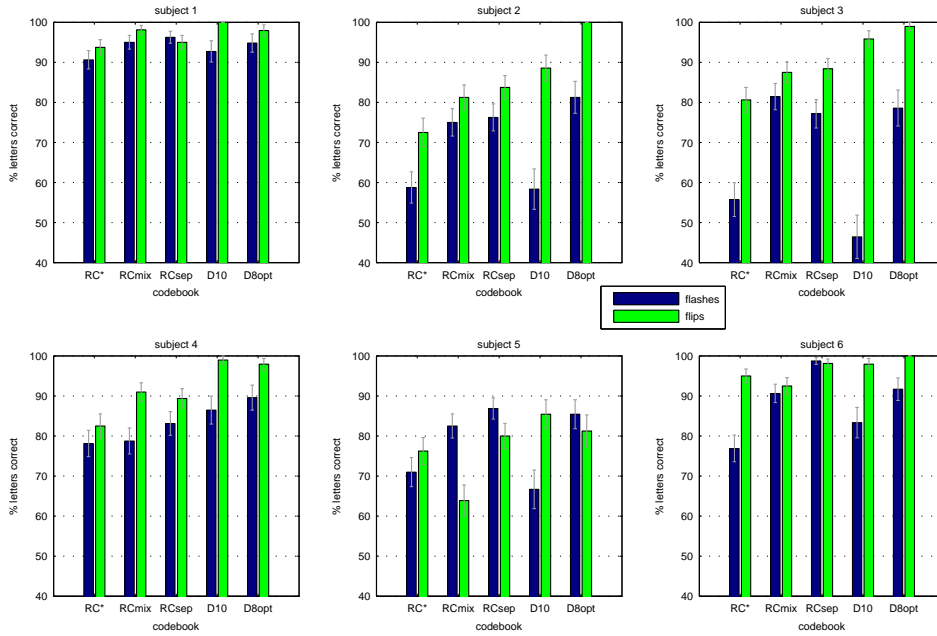

Figure 3: Offline (leave-one-letter-out) 36-class prediction performance when decoding codewords of length 24, for each of the subjects in each of the code conditions.

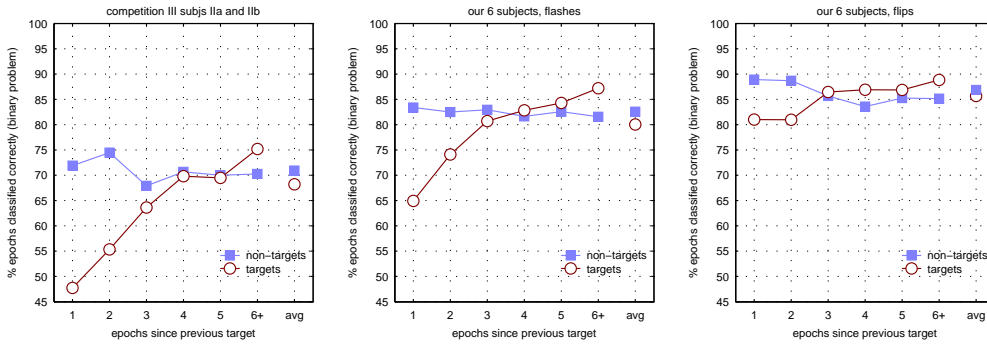

Figure 4: Illustration of effect of TPT on epoch classification performance, (left) in the data from competition III dataset II; (middle) in our experiments, averaged across all subjects and code conditions for blocks in which the flash stimulus was used; (right) in our experiments, averaged across the same subjects and code conditions, but for blocks in which the flip stimulus was used. The rightmost column of each plot shows average classification accuracy across all epochs (remember that short TTIs are relatively uncommon overall, and therefore downweighted in the average).

performing slightly less well than $\mathbf{RC}_{sep}$, which has longer TTIs. However, the latter effect is not as large or as consistent across subjects as it was in our preliminary study [7].

2. Using the Donchin flash stimulus, our optimized code $\mathbf{D8}_{opt}$ performs about as well as traditional $\mathbf{RC}$ codes, but does not outperform them.

3. Generally, performance using the flip stimulus is better than with the flash stimulus.

4. Using the flip stimulus, both $\mathbf{D8}_{opt}$ and $\mathbf{D10}$ perform better than the $\mathbf{RC}$ codes, and they perform roughly equally as well as each other. We interpret this interaction between stimulus type and code type as an indication that the flip stimulus may generate rather different psychophysiological responses from the flash (perhaps stronger primary visual evoked-potentials, in addition to the P300) of a kind which is less susceptible to short TTI (the

curves in the right panel of figure 4 being flatter than those in the middle panel). A comparative analysis of the spatial locations of discriminative sources in the two stimulus conditions is beyond the scope of the current short report.

5. Despite having identical TTIs and Hamming distances, $\mathbf{RC}_*$ performs consistently worse than $\mathbf{RC}_{\text{sep}}$, in both stimulus conditions.

In summary, we have obtained empirical support for the idea that TTI (finding #1), Hamming distance (finding #4) and stimulus type (finding #3) can all be manipulated to improve performance. However, our initial attempt to find an optimal solution by balancing these effects was not successful (finding #2). In the flash stimulus condition, the row-column codes performed better than expected, matching the performance of our optimized code. In the flip stimulus condition, TTI effects were greatly reduced, making either $\mathbf{D8}_{\text{opt}}$ or $\mathbf{D10}$ suitable despite the short TTIs of the latter.

It seems very likely that the unexpectedly high performance of $\mathbf{RC}_{\text{sep}}$ and $\mathbf{RC}_{\text{mix}}$ can be at least partly explained by the idea that they have particular *spatial* properties that enhance their performance beyond what Hamming distances and TTIs alone would predict. This hypothesis is corroborated by finding #5. Models of such spatial effects should clearly be taken into account in future optimization approaches.

Overall, best performance was obtained with the flip stimulus, using either of the two error-correcting codes, $\mathbf{D8}_{\text{opt}}$ or $\mathbf{D10}$: this consistently outperforms the traditional row-column flash design and shows that error-correcting code design has an important role to play in BCI speller development.

As a final note, one should remember that a language model can be used to improve performance in speller systems. In this case, the codebook optimization problem becomes more complicated than the simplified setting we examined, because the prior $\Pr(t)$ in (2) is no longer flat. The nature of the best codes, according to our optimization criterion, might change considerably: for example, a small subset of codewords, representing the most probable letters, might be chosen to be particularly sparse and/or to have a particularly large Hamming distance between them and between the rest of the codebook, while within the rest of the codebook these two criteria might be considered relatively unimportant. Ideally, the language model would be adaptive (for example, supplying a predictive prior for each letter based on the previous three) which might mean that the codewords should be reassigned optimally after each letter. However, such considerations must remain beyond the scope of our study until we can either overcome the TTI-independent performance differences between codes (perhaps, as our results suggest, by careful stimulus design), or until we can model the source of these differences well enough to account for them in our optimization criterion.

# References

[1] Bishop CM (1995) Neural Networks for Pattern Recognition. *Clarendon Press, Oxford*.

[2] Blankertz B, *et al*. (2006) *IEEE Trans. Neural Systems & Rehab. Eng.* **14**(2): 153–159

[3] Donchin E, Coles MGH (1988) *Behavioural and Brain Sciences* **11**: 357–374

[4] Farwell LA, Donchin E (1988) *Electroencephalography and Clinical Neurophysiology* **70**: 510–523

[5] Gestel T, *et al*. (2002) *Neural Processing Letters*, **15**: 45–48

[6] Gonsalvez CL, Polich J (2002) *Psychophysiology* **39**(3): 388–96

[7] Hill NJ, *et al* (2008) Technical Report #166, Max Planck Institute for Biological Cybernetics.

[8] Krusienski DJ, *et al*. (2006) *Journal of Neural Engineering* **3**(4): 299–305

[9] MacKay D (2005) Information Theory, Inference, and Learning Algorithms. *Cambridge Univ. Press*

[10] Martens SMM, Hill NJ, Farquhar J, Schölkopf B. (2007) Impact of Target-to-Target Interval on Classification Performance in the P300 Speller. *Applied Neuroscience Conference*, Nijmegen, The Netherlands.

[11] Pritchard WS (1981) *Psychological Bulletin* **89**: 506–540

[12] Rugg MD, Coles MGH (2002) Electrophysiology of mind. *Oxford Psychology Series 25*

[13] Serby H, Yom-Tov E, Inbar GF (2005) *IEEE Trans. Neural Systems & Rehab. Eng.* **13**(1):89-98

[14] Wolpaw JR, *et al*. (2002) *Clinical Neurophysiology* **113**: 767–791

[15] Woods DL, Hillyard SA, Courchesne E, Galambos R. (1980) *Science*, New Series **207**(4431): 655–657.

